# The Infinite Gaussian Mixture Model

**Carl Edward Rasmussen**
Department of Mathematical Modelling
Technical University of Denmark
Building 321, DK-2800 Kongens Lyngby, Denmark
carl@imm.dtu.dk http://bayes.imm.dtu.dk

## Abstract

In a Bayesian mixture model it is not necessary a priori to limit the number of components to be finite. In this paper an infinite Gaussian mixture model is presented which neatly sidesteps the difficult problem of finding the "right" number of mixture components. Inference in the model is done using an efficient parameter-free Markov Chain that relies entirely on Gibbs sampling.

## 1 Introduction

One of the major advantages in the Bayesian methodology is that "overfitting" is avoided; thus the difficult task of adjusting model complexity vanishes. For neural networks, this was demonstrated by Neal [1996] whose work on infinite networks led to the reinvention and popularisation of Gaussian Process models [Williams & Rasmussen, 1996]. In this paper a Markov Chain Monte Carlo (MCMC) implementation of a hierarchical infinite Gaussian mixture model is presented. Perhaps surprisingly, inference in such models is possible using finite amounts of computation.

Similar models are known in statistics as Dirichlet Process mixture models and go back to Ferguson [1973] and Antoniak [1974]. Usually, expositions start from the Dirichlet process itself [West et al, 1994]; here we derive the model as the limiting case of the well-known finite mixtures. Bayesian methods for mixtures with an unknown (finite) number of components have been explored by Richardson & Green [1997], whose methods are not easily extended to multivariate observations.

## 2 Finite hierarchical mixture

The finite Gaussian mixture model with $k$ components may be written as:

$$p(y|\mu_1,\ldots,\mu_k,s_1,\ldots,s_k,\pi_1,\ldots,\pi_k) = \sum_{j=1}^{k} \pi_j \mathcal{N}\left(\mu_j, s_j^{-1}\right), \qquad (1)$$

where $\mu_j$ are the means, $s_j$ the *precisions* (inverse variances), $\pi_j$ the mixing proportions (which must be positive and sum to one) and $\mathcal{N}$ is a (normalised) Gaussian with specified mean and variance. For simplicity, the exposition will initially assume scalar observations, $n$ of which comprise the training data $\mathbf{y} = \{y_1,\ldots,y_n\}$. First we will consider these models for a fixed value of $k$, and later explore the properties in the limit where $k \to \infty$.

Gibbs sampling is a well known technique for generating samples from complicated multivariate distributions that is often used in Monte Carlo procedures. In its simplest form, Gibbs sampling is used to update each variable in turn from its conditional distribution given all other variables in the system. It can be shown that Gibbs sampling generates samples from the joint distribution, and that the entire distribution is explored as the number of Gibbs sweeps grows large.

We introduce stochastic *indicator* variables, $c_i$, one for each observation, whose role is to encode which class has generated the observation; the indicators take on values $1 \ldots k$. Indicators are often referred to as "missing data" in a mixture model context.

In the following sections the priors on component parameters and hyperparameters will be specified, and the conditional distributions for these, which will be needed for Gibbs sampling, will be derived. In general the form of the priors are chosen to have (hopefully) reasonable modelling properties, with an eye to mathematical convenience (through the use of conjugate priors).

## 2.1 Component parameters

The component means, $\mu_j$, are given Gaussian priors:

$$p(\mu_j | \lambda, r) \sim \mathcal{N}(\lambda, r^{-1}), \tag{2}$$

whose mean, $\lambda$, and precision, $r$, are hyperparameters common to all components. The hyperparameters themselves are given vague Normal and Gamma priors:

$$p(\lambda) \sim \mathcal{N}(\mu_y, \sigma_y^2), \qquad p(r) \sim \mathcal{G}(1, \sigma_y^{-2}) \propto r^{-1/2} \exp(-r\sigma_y^2/2), \tag{3}$$

where $\mu_y$ and $\sigma_y^2$ are the mean and variance of the observations[1]. The shape parameter of the Gamma prior is set to unity, corresponding to a very broad (vague) distribution.

The conditional posterior distributions for the means are obtained by multiplying the likelihood from eq. (1) conditioned on the indicators, by the prior, eq. (2):

$$p(\mu_j | \mathbf{c}, \mathbf{y}, s_j, \lambda, r) \sim \mathcal{N}\Big(\frac{\bar{y}_j n_j s_j + \lambda r}{n_j s_j + r}, \frac{1}{n_j s_j + r}\Big), \qquad \bar{y}_j = \frac{1}{n_j} \sum_{i:c_i=j} y_i, \tag{4}$$

where the *occupation number*, $n_j$, is the number of observations belonging to class $j$, and $\bar{y}_j$ is the mean of these observations. For the hyperparameters, eq. (2) plays the role of the likelihood which together with the priors from eq. (4) give conditional posteriors of standard form:

$$p(\lambda | \mu_1, \ldots, \mu_k, r) \sim \mathcal{N}\Big(\frac{\mu_y \sigma_y^{-2} + r \sum_{j=1}^{k} \mu_j}{\sigma_y^{-2} + kr}, \frac{1}{\sigma_y^{-2} + kr}\Big),$$

$$p(r | \mu_1, \ldots, \mu_k, \lambda) \sim \mathcal{G}\Big(k+1, \big[\frac{1}{k+1}(\sigma_y^2 + \sum_{j=1}^{k}(\mu_j - \lambda)^2)\big]^{-1}\Big). \tag{5}$$

The component precisions, $s_j$, are given Gamma priors:

$$p(s_j | \beta, w) \sim \mathcal{G}(\beta, w^{-1}), \tag{6}$$

whose shape, $\beta$, and mean, $w^{-1}$, are hyperparameters common to all components, with priors of inverse Gamma and Gamma form:

$$p(\beta^{-1}) \sim \mathcal{G}(1, 1) \implies p(\beta) \propto \beta^{-3/2} \exp(-1/(2\beta)), \qquad p(w) \sim \mathcal{G}(1, \sigma_y^2). \tag{7}$$

The conditional posterior precisions are obtained by multiplying the likelihood from eq. (1) conditioned on the indicators, by the prior, eq. (6):

$$p(s_j|\mathbf{c},\mathbf{y},\mu_j,\beta,w) \sim \mathcal{G}\Big(\beta + n_j, \big[\frac{1}{\beta+n_j}(w\beta + \sum_{i:c_i=j}(y_i-\mu_j)^2)\big]^{-1}\Big). \qquad (8)$$

For the hyperparameters, eq. (6) plays the role of likelihood which together with the priors from eq. (7) give:

$$p(w|s_1,\ldots,s_k,\beta) \sim \mathcal{G}\Big(k\beta+1, \big[\frac{1}{k\beta+1}(\sigma_y^{-2} + \beta\sum_{j=1}^{k}s_j)\big]^{-1}\Big), \qquad (9)$$

$$p(\beta|s_1,\ldots,s_k,w) \propto \Gamma\big(\frac{\beta}{2}\big)^{-k}\exp\big(\frac{-1}{2\beta}\big)\big(\frac{\beta}{2}\big)^{(k\beta-3)/2}\prod_{j=1}^{k}(s_jw)^{\beta/2}\exp\big(-\frac{\beta s_j w}{2}\big).$$

The latter density is not of standard form, but it can be shown that $p(\log(\beta)|s_1,\ldots,s_k,w)$ is log-concave, so we may generate independent samples from the distribution for $\log(\beta)$ using the Adaptive Rejection Sampling (ARS) technique [Gilks & Wild, 1992], and transform these to get values for $\beta$.

The mixing proportions, $\pi_j$, are given a symmetric Dirichlet (also known as multivariate beta) prior with concentration parameter $\alpha/k$:

$$p(\pi_1,\ldots,\pi_k|\alpha) \sim \text{Dirichlet}(\alpha/k,\ldots,\alpha/k) = \frac{\Gamma(\alpha)}{\Gamma(\alpha/k)^k}\prod_{j=1}^{k}\pi_j^{\alpha/k-1}, \qquad (10)$$

where the mixing proportions must be positive and sum to one. Given the mixing proportions, the prior for the occupation numbers, $n_j$, is multinomial and the joint distribution of the indicators becomes:

$$p(c_1,\ldots,c_k|\pi_1,\ldots,\pi_k) = \prod_{j=1}^{k}\pi_j^{n_j}, \qquad n_j = \sum_{i=1}^{n}\delta_{\text{Kronecker}}(c_i,j). \qquad (11)$$

Using the standard Dirichlet integral, we may integrate out the mixing proportions and write the prior directly in terms of the indicators:

$$p(c_1,\ldots,c_k|\alpha) = \int p(c_1,\ldots,c_k|\pi_1,\ldots,\pi_k)p(\pi_1,\ldots,\pi_k)d\pi_1\cdots d\pi_k \qquad (12)$$

$$= \frac{\Gamma(\alpha)}{\Gamma(\alpha/k)^k}\int \prod_{j=1}^{k}\pi_j^{n_j+\alpha/k-1}d\pi_j = \frac{\Gamma(\alpha)}{\Gamma(n+\alpha)}\prod_{j=1}^{k}\frac{\Gamma(n_j+\alpha/k)}{\Gamma(\alpha/k)}.$$

In order to be able to use Gibbs sampling for the (discrete) indicators, $c_i$, we need the conditional prior for a single indicator given all the others; this is easily obtained from eq. (12) by keeping all but a single indicator fixed:

$$p(c_i = j|\mathbf{c}_{-i},\alpha) = \frac{n_{-i,j}+\alpha/k}{n-1+\alpha}, \qquad (13)$$

where the subscript $-i$ indicates all indexes except $i$ and $n_{-i,j}$ is the number of observations, excluding $y_i$, that are associated with component $j$. The posteriors for the indicators are derived in the next section.

Lastly, a vague prior of inverse Gamma shape is put on the concentration parameter $\alpha$:

$$p(\alpha^{-1}) \sim \mathcal{G}(1,1) \implies p(\alpha) \propto \alpha^{-3/2}\exp\big(-1/(2\alpha)\big). \qquad (14)$$

The likelihood for $\alpha$ may be derived from eq. (12), which together with the prior from eq. (14) gives:

$$p(n_1,\dots,n_k|\alpha) = \frac{\alpha^k \Gamma(\alpha)}{\Gamma(n+\alpha)}, \qquad p(\alpha|k,n) \propto \frac{\alpha^{k-3/2}\exp\big(-1/(2\alpha)\big)\Gamma(\alpha)}{\Gamma(n+\alpha)}. \quad (15)$$

Notice, that the conditional posterior for $\alpha$ depends only on number of observations, $n$, and the number of components, $k$, and not on how the observations are distributed among the components. The distribution $p(\log(\alpha)|k,n)$ is log-concave, so we may efficiently generate independent samples from this distribution using ARS.

## 3 The infinite limit

So far, we have considered $k$ to be a fixed finite quantity. In this section we will explore the limit $k \to \infty$ and make the final derivations regarding the conditional posteriors for the indicators. For all the model variables except the indicators, the conditional posteriors for the infinite limit is obtained by substituting for $k$ the number of classes that have data associated with them, $k_{\text{rep}}$, in the equations previously derived for the finite model. For the indicators, letting $k \to \infty$ in eq. (13), the conditional prior reaches the following limits:

components where $n_{-i,j} > 0$: $\qquad p(c_i = j|\mathbf{c}_{-i},\alpha) \ = \ \dfrac{n_{-i,j}}{n-1+\alpha},$

all other components combined: $\quad p(c_i \neq c_{i'} \text{ for all } i' \neq i|\mathbf{c}_{-i},\alpha) \ = \ \dfrac{\alpha}{n-1+\alpha}.$ $\qquad(16)$

This shows that the conditional class prior for components that are associated with other observations is proportional to the number of such observations; the combined prior for all other classes depends only on $\alpha$ and $n$. Notice how the analytical tractability of the integral in eq. (12) is essential, since it allows us to work directly with the (finite number of) indicator variables, rather than the (infinite number of) mixing proportions. We may now combine the likelihood from eq. (1) conditioned on the indicators with the prior from eq. (16) to obtain the conditional posteriors for the indicators:

components for which $n_{-i,j} > 0$: $\quad p(c_i = j|\mathbf{c}_{-i},\mu_j,s_j,\alpha) \propto$ $\qquad\qquad\qquad\qquad(17)$

$$p(c_i = j|\mathbf{c}_{-i},\alpha)p(y_i|\mu_j,s_j,\mathbf{c}_{-i}) \propto \frac{n_{-i,j}}{n-1+\alpha}s_j^{1/2}\exp\big(-s_j(y_i-\mu_j)^2/2\big),$$

all other components combined: $\quad p(c_i \neq c_{i'} \text{ for all } i \neq i'|\mathbf{c}_{-i},\lambda,r,\beta,w,\alpha) \propto$

$$p(c_i \neq c_{i'} \text{ for all } i \neq i'|\mathbf{c}_{-i},\alpha)\int p(y_i|\mu_j,s_j)p(\mu_j,s_j|\lambda,r,\beta,w)d\mu_j ds_j.$$

The likelihood for components with observations other than $y_i$ currently associated with them is Gaussian with component parameters $\mu_j$ and $s_j$. The likelihood pertaining to the currently unrepresented classes (which have no parameters associated with them) is obtained through integration over the prior distribution for these. Note, that we need not differentiate between the infinitely many unrepresented classes, since their parameter distributions are all identical. Unfortunately, this integral is not analytically tractable; I follow Neal [1998], who suggests to sample from the priors (which are Gaussian and Gamma shaped) in order to generate a Monte Carlo estimate of the probability of "generating a new class". Notice, that this approach effectively generates parameters (by sampling from the prior) for the classes that are unrepresented. Since this Monte Carlo estimate is unbiased, the resulting chain will sample from *exactly* the desired distribution, no matter how many samples are used to approximate the integral; I have found that using a single sample works fairly well in many applications.

In detail, there are three possibilities when computing conditional posterior class probabilities, depending on the number of observations associated with the class:

**if** $n_{-i,j} > 0$: there are other observations associated with class $j$, and the posterior class probability is as given by the top line of eq. (17).

**if** $n_{-i,j} = 0$ **and** $c_i = j$: observation $y_i$ is currently the only observation associated with class $j$; this is an peculiar situation, since there are no other observations associated with the class, but the class still has parameters. It turns out that this situation should be handled as an unrepresented class, but rather than sampling for the parameters, one simply uses the class parameters; consult [Neal 1998] for a detailed derivation.

**unrepresented classes:** values for the mixture parameters are picked at random from the prior for these parameters, which is Gaussian for $\mu_j$ and Gamma shaped for $s_j$.

Now that all classes have parameters associated with them, we can easily evaluate their likelihoods (which are Gaussian) and the priors, which take the form $n_{-i,j}/(n-1+\alpha)$ for components with observations other than $y_i$ associated with them, and $\alpha/(n-1+\alpha)$ for the remaining class. When hitherto unrepresented classes are chosen, a new class is introduced in the model; classes are removed when they become empty.

## 4  Inference; the "spirals" example

To illustrate the model, we use the 3 dimensional "spirals" dataset from [Ueda et al, 1998], containing 800 data point, plotted in figure 1. Five data points are generated from each of 160 isotropic Gaussians, whose means follow a spiral pattern.

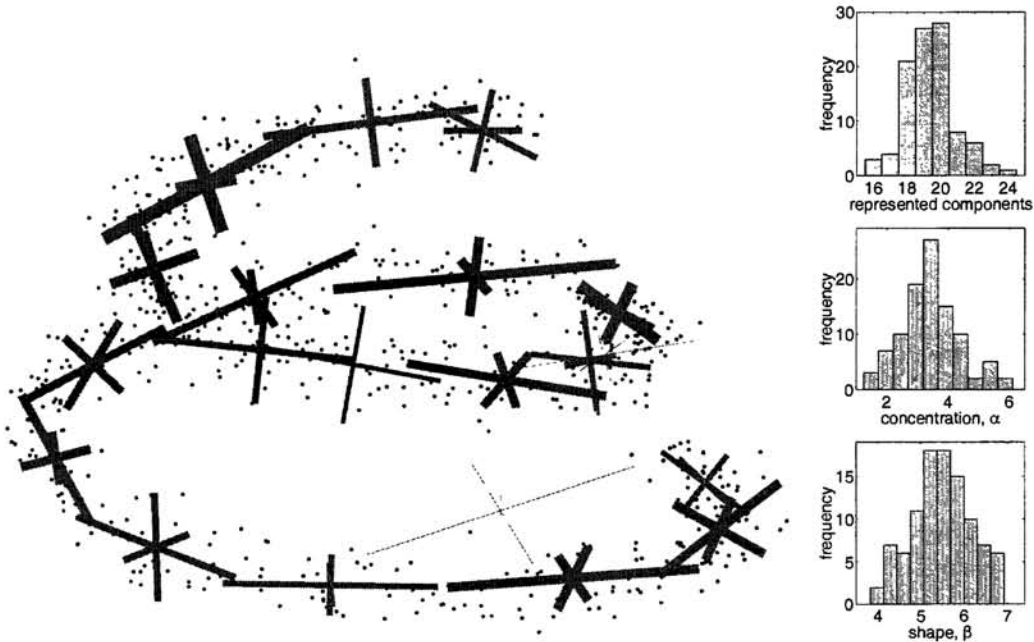

Figure 1: The 800 cases from the three dimensional spirals data. The crosses represent a single (random) sample from the posterior for the mixture model. The $k_{rep} = 20$ represented classes account for $n/(n+\alpha) \simeq 99.6\%$ of the mass. The lines indicate 2 std. dev. in the Gaussian mixture components; the thickness of the lines represent the mass of the class. To the right histograms for 100 samples from the posterior for $k_{rep}$, $\alpha$ and $\beta$ are shown.

### 4.1  Multivariate generalisation

The generalisation to multivariate observations is straightforward. The means, $\mu_j$, and precisions, $s_j$, become vectors and matrices respectively, and their prior (and posterior)

distributions become multivariate Gaussian and Wishart. Similarly, the hyperparameter $\lambda$ becomes a vector (multivariate Gaussian prior) and $r$ and $w$ become matrices with Wishart priors. The $\beta$ parameter stays scalar, with the prior on $(\beta - D + 1)^{-1}$ being Gamma with mean $1/D$, where $D$ is the dimension of the dataset. All other specifications stay the same. Setting $D = 1$ recovers the scalar case discussed in detail.

## 4.2  Inference

The mixture model is started with a single component, and a large number of Gibbs sweeps are performed, updating all parameters and hyperparameters in turn by sampling from the conditional distributions derived in the previous sections. In figure 2 the auto-covariance for several quantities is plotted, which reveals a maximum correlation-length of about 270. Then 30000 iterations are performed for modelling purposes (taking 18 minutes of CPU time on a Pentium PC): 3000 steps initially for "burn-in", followed by 27000 to generate 100 roughly independent samples from the posterior (spaced evenly 270 apart). In figure 1, the represented components of one sample from the posterior is visualised with the data. To the right of figure 1 we see that the posterior number of represented classes is very concentrated around $18 - 20$, and the concentration parameter takes values around $\alpha \simeq 3.5$ corresponding to only $\alpha/(n+\alpha) \simeq 0.4\%$ of the mass of the predictive distribution belonging to unrepresented classes. The shape parameter $\beta$ takes values around $5-6$, which gives the "effective number of points" contributed from the prior to the covariance matrices of the mixture components.

## 4.3  The predictive distribution

Given a particular state in the Markov Chain, the predictive distribution has two parts: the represented classes (which are Gaussian) and the unrepresented classes. As when updating the indicators, we may chose to approximate the unrepresented classes by a finite mixture of Gaussians, whose parameters are drawn from the prior. The final predictive distribution is an average over the (eg. 100) samples from the posterior. For the spirals data this density has roughly 1900 components for the represented classes plus however many are used to represent the remaining mass. I have not attempted to show this distribution. However, one can imagine a smoothed version of the single sample shown in figure 1, from averaging over models with slightly varying numbers of classes and parameters. The (small) mass from the unrepresented classes spreads diffusely over the entire observation range.

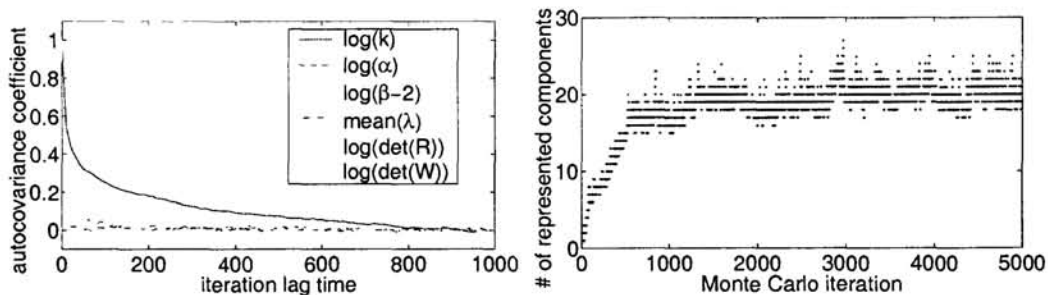

Figure 2: The left plot shows the auto-covariance length for various parameters in the Markov Chain, based on $10^5$ iterations. Only the number of represented classes, $k_{\mathrm{rep}}$, has a significant correlation; the effective correlation length is approximately 270, computed as the sum of covariance coefficients between lag $-1000$ and 1000. The right hand plot shows the number of represented classes growing during the initial phase of sampling. The initial 3000 iterations are discarded.

# 5  Conclusions

The infinite hierarchical Bayesian mixture model has been reviewed and extended into a practical method. It has been shown that good performance (without overfitting) can be achieved on multidimensional data. An efficient and practical MCMC algorithm with no free parameters has been derived and demonstrated on an example. The model is fully automatic, without needing specification of parameters of the (vague) prior. This corroborates the falsity of the common misconception that "the only difference between Bayesian and non-Bayesian methods is the prior, which is arbitrary anyway ... ".

Further tests on a variety of problems reveals that the infinite mixture model produces densities whose generalisation is highly competitive with other commonly used methods. Current work is undertaken to explore performance on high dimensional problems, in terms of computational efficiency and generalisation.

The infinite mixture model has several advantages over its finite counterpart: 1) in many applications, it may be more appropriate not to limit the number of classes, 2) the number of represented classes is automatically determined, 3) the use of MCMC effectively avoids local minima which plague mixtures trained by optimisation based methods, eg. EM [Ueda et al, 1998] and 4) it is much simpler to handle the infinite limit than to work with finite models with unknown sizes, as in [Richardson & Green, 1997] or traditional approaches based on extensive crossvalidation. The Bayesian infinite mixture model solves simultaneously several long-standing problems with mixture models for density estimation.

## Acknowledgments

Thanks to Radford Neal for helpful comments, and to Naonori Ueda for making the spirals data available. This work is funded by the Danish Research Councils through the Computational Neural Network Center (CONNECT) and the THOR Center for Neuroinformatics.

## Footnotes

[1]Strictly speaking, the priors ought not to depend on the observations. The current procedure is equivalent to normalising the observations and using unit priors. A wide variety of reasonable priors will lead to similar results.

## References

Antoniak, C. E. (1974). Mixtures of Dirichlet processes with applications to Bayesian nonparametric problems. *Annals of Statistics 2*, 1152–1174.

Ferguson, T. S. (1973). A Bayesian analysis of some nonparametric problems. *Annals of Statistics 1*, 209–230.

Gilks, W. R. and P. Wild (1992). Adaptive rejection sampling for Gibbs sampling. *Applied Statistics 41*, 337–348.

Neal, R. M. (1996). Bayesian Learning for Neural Networks, Lecture Notes in Statistics No. 118, New York: Springer-Verlag.

Neal, R. M. (1998). Markov chain sampling methods for Dirichlet process mixture models. Technical Report 4915, Department of Statistics, University of Toronto. http://www.cs.toronto.edu/~radford/mixmc.abstract.html.

Richardson, S. and P. Green (1997). On Bayesian analysis of mixtures with an unknown number of components. *Journal of the Royal Statistical Society, B 59*, 731–792.

Ueda, N., R. Nakano, Z. Ghahramani and G. E. Hinton (1998). SMEM Algorithm for Mixture Models, NIPS 11, MIT Press.

West, M., P. Müller and M. D. Escobar (1994). Hierarchical priors and mixture models with applications in regression and density estimation. In P. R. Freeman and A. F. M. Smith (editors), *Aspects of Uncertainty*, pp. 363–386. John Wiley.

Williams, C. K. I. and C. E. Rasmussen (1996). Gaussian Processes for Regression, in D. S. Touretzky, M. C. Mozer and M. E. Hasselmo (editors), NIPS 8, MIT Press.